# Fractional Belief Propagation

**Wim Wiegerinck and Tom Heskes**
SNN, University of Nijmegen
Geert Grooteplein 21, 6525 EZ, Nijmegen, the Netherlands
{wimw,tom}@snn.kun.nl

## Abstract

We consider loopy belief propagation for approximate inference in probabilistic graphical models. A limitation of the standard algorithm is that clique marginals are computed as if there were no loops in the graph. To overcome this limitation, we introduce fractional belief propagation. Fractional belief propagation is formulated in terms of a family of approximate free energies, which includes the Bethe free energy and the naive mean-field free as special cases. Using the linear response correction of the clique marginals, the scale parameters can be tuned. Simulation results illustrate the potential merits of the approach.

## 1  Introduction

Probabilistic graphical models are powerful tools for learning and reasoning in domains with uncertainty. Unfortunately, inference in large, complex graphical models is computationally intractable. Therefore, approximate inference methods are needed. Basically, one can distinguish between to types of methods, stochastic sampling methods and deterministic methods. One of methods in the latter class is Pearl's loopy belief propagation [1]. This method is increasingly gaining interest since its successful applications to turbo-codes. Until recently, a disadvantage of the method was its heuristic character, and the absence of a converge guarantee. Often, the algorithm gives good solutions, but sometimes the algorithm fails to converge. However, Yedidia et al. [2] showed that the fixed points of loopy belief propagation are actually stationary points of the Bethe free energy from statistical physics. This does not only give the algorithm a firm theoretical basis, but it also solves the convergence problem by the existence of an objective function which can be minimized directly [3]. Belief propagation is generalized in several directions. Minka's expectation propagation [4] is a generalization that makes the method applicable to Bayesian learning. Yedidia et al. [2] introduced the Kikuchi free energy in the graphical models community, which can be considered as a higher order truncation of a systematic expansion of the exact free energy using larger clusters. They also developed an associated generalized belief propagation algorithm. In this paper, we propose another direction which yields possibilities to improve upon loopy belief propagation, without resorting to larger clusters.

This paper is organized as follows. In section 2 we define the inference problem. In section 3 we shortly review approximate inference by loopy belief propagation and discuss an inherent limitation of this method. This motivates us to generalize upon loopy belief propagation. We do so by formulating a new class of approximate free energies in section 4. In

section 5 we consider the fixed point equations and formulate the fractional belief propagation algorithm. In section 6 we will use linear response estimates to tune the parameters in the method. Simulation results are presented in section 7. In section 8 we end with the conclusion.

## 2 Inference in graphical models

Our starting point is a probabilistic model $P$ on a set of discrete variables $x = x_1, \ldots, x_n$ in a finite domain. The joint distribution $P(x)$ is assumed to be proportional to a product of clique potentials

$$P(x) \propto \prod_\alpha \psi_\alpha(x_\alpha) \,, \tag{1}$$

where each $\alpha$ refers to a subset of the $n$ nodes in the model. A typical example that we will consider later in the paper is the Boltzmann machine with binary units ($x_i = \pm 1$),

$$P(x) \propto \exp(\sum_{(i,j)} w_{ij} x_i x_j + \sum_k \theta_k x_k) \,, \tag{2}$$

where the sum is over connected pairs $(i, j)$. The right hand side can be viewed as product of potentials $\psi_{ij}(x_{ij}) = \exp(w_{ij} x_i x_j + \frac{1}{(|N_i|)} \theta_i x_i + \frac{1}{|N_j|} \theta_j x_j))$, where $N_i$ is the set of edges that contain node $i$. The typical task that we try to perform is to compute the marginal single node distributions $P(x_i)$. Basically, the computation requires the summation over all remaining variables $x \backslash x_i$. In small networks, this summation can be performed explicitly. In large networks, the complexity of computation depends on the underlying graphical structure of the model, and is exponential in the maximal clique size of the triangulated moralized graph [5]. This may lead to intractable models, even if the clusters $x_\alpha$ are small. When the model is intractable, one has to resort to approximate methods.

## 3 Loopy belief propagation in Boltzmann machines

A nowadays popular approximate method is loopy belief propagation. In this section, we will shortly review of this method. Next we will discuss one of its inherent limitations, which motivates us to propose a possible way to overcome this limitation. For simplicity, we restrict this section to Boltzmann machines.

The goal is to compute pair marginals $P(x_{ij})$ of connected nodes. Loopy belief propagation computes approximating pair marginals $Q_{ij}(x_{ij})$ by applying the belief propagation algorithm for trees to loopy graphs, i.e., it computes messages according to

$$m_{i \to j}(x_j) \propto \sum_{x_i} \exp(w_{ij} x_i x_j) \hat\phi_{ij}(x_i) \,, \tag{3}$$

in which $\hat\phi_{ij}$ are the incoming messages to node $i$ except from node $j$,

$$\hat\phi_{ij}(x_i) = \exp(\theta_i x_i) \prod_{k \in N_i \backslash j} m_{k \to i}(x_i) \,. \tag{4}$$

If the procedure converges (which is not guaranteed in loopy graphs), the resulting approximating pair marginals are

$$Q_{ij}(x_{ij}) \propto \exp(w_{ij} x_i x_j) \hat\phi_{ij}(x_i) \hat\phi_{ji}(x_j) \,. \tag{5}$$

In general, the exact pair marginals will be of the form

$$P_{ij}(x_{ij}) \propto \exp(w_{ij}^{\text{eff}} x_i x_j) \phi_{ij}(x_i) \phi_{ji}(x_j) \,, \tag{6}$$

which has an effective interaction $w_{ij}^{\text{eff}}$. In the case of a tree, $w_{ij}^{\text{eff}} = w_{ij}$. With loops in the graph, however, the loops will contribute to $w_{ij}^{\text{eff}}$, and the result will in general be different from $w_{ij}$. If we compare (6) with (5), we see that loopy belief propagation assumes $w_{ij}^{\text{eff}} = w_{ij}$, ignoring contributions from loops.

Now suppose we would know $w_{ij}^{\text{eff}}$ in advance, then a better approximation could be expected if we could model approximate pair marginals of the form

$$Q_{ij}(x_{ij}) \propto \exp(\frac{w_{ij}}{c_{ij}} x_i x_j) \hat{\phi}_{ij}(x_i) \hat{\phi}_{ji}(x_j) \ , \tag{7}$$

where $c_{ij} = w_{ij}/w_{ij}^{\text{eff}}$. The $\hat{\phi}_{ij}$ are to be determined by some propagation algorithm.

In the next sections, we generalize upon the above idea and introduce fractional belief propagation as a family of loopy belief propagation-like algorithms parameterized by scale parameters $\vec{c} = \{c_\alpha\}$. The resulting approximating clique marginals will be of the form

$$Q_\alpha(x_\alpha) \propto \psi_\alpha(x_\alpha)^{1/c_\alpha} \prod_{i \in N_\alpha} \hat{\phi}_i(x_i) \ , \tag{8}$$

where $N_\alpha$ is the set of nodes in clique $\alpha$. The issue of how to set the parameters $\vec{c}$ is subject of section 6.

## 4 A family of approximate free energies

The new class of approximating methods will be formulated via a new class of approximating free energies. The exact free energy of a model with clique potentials $\{\psi_\alpha\}$ is

$$F_{\{\psi_\alpha\}}(P) \equiv - \sum_x P(x) \sum_\alpha \log \psi_\alpha(x_\alpha) + \sum_x P(x) \log P(x) \ . \tag{9}$$

It is well known that the joint distribution $P$ can be recovered by minimization of the free energy

$$P = \operatorname*{argmin}_{\hat{P}} F_{\{\psi_\alpha\}}(\hat{P}) \tag{10}$$

under the constraint $\sum_x P(x) = 1$. The idea is now to construct an approximate free energy $F^{\text{Approx}}(\hat{Q})$ and compute its minimum $Q$. Then $Q$ is interpreted as an approximation of $P$.

A popular approximate free energy is based on the Bethe assumption, which basically states that $Q$ is approximately tree-like,

$$Q(x) = \prod_\alpha Q_\alpha(x_\alpha) \prod_i Q_i(x_i)^{1 - |N_i|} \ , \tag{11}$$

in which $N_i$ are the cliques $\alpha$ that contain $i$. This assumption is exact if the factor graph [6] of the model is a tree. Substitution of the tree-assumption into the free energy leads to the well-known Bethe free energy

$$F_{\{\psi_\alpha\}}^{\text{Bethe}}(\{Q_\alpha, Q_i\}) = - \sum_\alpha \sum_{x_\alpha} Q_\alpha(x_\alpha) \log \psi_\alpha(x_\alpha)$$

$$+ \sum_\alpha \sum_{x_\alpha} Q_\alpha(x_\alpha) \log Q_\alpha(x_\alpha) + \sum_i (1 - |N_i|) \sum_{x_i} Q_i(x_i) \log Q_i(x_i) \ , \tag{12}$$

which is to be minimized under normalization constraints $\sum_{x_\alpha} Q_\alpha(x_\alpha) = 1$ and $\sum_{x_i} Q_i(x_i) = 1$ and the marginalization constraints $\sum_{x_{\alpha \setminus i}} Q_\alpha(x_\alpha) = Q_i(x_i)$ for $i \in N_\alpha$.

It can be shown that minima of the Bethe free energy are fixed points of the loopy belief propagation algorithm [2].

In our proposal, we generalize upon the Bethe assumption, and make the parameterized assumption

$$Q(x) = \prod_\alpha Q_\alpha(x_\alpha)^{c_\alpha} \prod_i Q_i(x_i)^{1-c_i|N_i|} , \qquad (13)$$

in which $c_i = 1/|N_i| \sum_{\alpha \in N_i} c_\alpha$. The intuition behind this assumption is that we replace each $\psi_\alpha(x_\alpha)$ by a factor $Q_\alpha(x_\alpha)^{c_\alpha}$. The term with single node marginals is constructed to deal with overcounted terms. Substitution of (13) into the free energy leads to the approximate free energy

$$F^{\vec{c}}_{\{\psi_\alpha\}}(\{Q_\alpha, Q_i\}) = - \sum_\alpha \sum_{x_\alpha} Q_\alpha(x_\alpha) \log \psi_\alpha(x_\alpha)$$
$$+ \sum_\alpha c_\alpha \sum_{x_\alpha} Q_\alpha(x_\alpha) \log Q(x_\alpha) + \sum_i (1 - c_i|N_i|) \sum_{x_i} Q_i(x_i) \log Q_i(x_i) , \quad (14)$$

which is also parameterized by $\vec{c}$. This class of free energies trivially contains the Bethe free energy ($c_\alpha = 1$). In addition, it includes the variational mean field free energy, conventionally written as $F^{\mathrm{MF}} = - \sum_\alpha \sum_{x_\alpha} \prod_i Q_i \log \psi_\alpha + \sum_i \sum_{x_i} Q_i \log Q_i$ as a limiting case for $c_\alpha = c \to \infty$ (implying an effective interaction of strength zero). If this limit is taken in (14), terms linear in $c$ will dominate and act as a penalty term for non-factorial entropies. Consequently, the distributions will be constrained to be completely factorized, $Q_\alpha = \prod_{i \in N_\alpha} Q_i$. Under these constraints, the remaining terms reduce to the conventional representation of $F^{\mathrm{MF}}$. Thirdly, it contains the recently derived free energy to upper bound the log partition function [7]. This one is recovered if, for pair-wise cliques, the $c_{ij}$'s are set to the edge appearance probabilities in the so-called spanning tree polytope of the graph. These requirements imply that $0 \leq c_{ij} \leq 1$.

## 5  Fractional belief propagation

In this section we will use the fixed point equations to generalize Pearl's algorithm to fractional belief propagation as a heuristic to minimize $F^{\vec{c}}$. Here, we do not worry too much about guaranteed convergence. If convergence is a problem, one can always resort to direct minimization of $F^{\vec{c}}$ using, e.g., Yuille's CCCP algorithm [3]. If standard belief propagation converges, its solution is guaranteed to be a local minimum of $F^{\mathrm{Bethe}}$ [8]. We expect a similar situation for $F^{\vec{c}}$.

Fixed point equations from $F^{\vec{c}}$ are derived in the same way as in [2]. We obtain

$$Q_\alpha(x_\alpha) \;\propto\; \psi_\alpha(x_\alpha)^{1/c_\alpha} \prod_{i \in N_\alpha} \prod_{\beta \in N_i \backslash \alpha} m_{\beta i}(x_i) m_{\alpha i}(x_i)^{1-1/c_\alpha} , \qquad (15)$$

$$Q_i(x_i) \;\propto\; \prod_\alpha m_{\alpha i}(x_i) , \qquad (16)$$

$$m_{\alpha i}(x_i) \;=\; \frac{Q_\alpha(x_i)}{Q_i(x_i)} m_{\alpha i}(x_i) . \qquad (17)$$

and we notice that $Q_\alpha(x_\alpha)$ has indeed the functional dependency of $\psi_\alpha$ as desired in (8). Inspired by Pearl's loopy belief propagation algorithm, we use the above equations to formulate fractional belief propagation $BP(\vec{c})$ (see Algorithm 1) [1].

**Algorithm 1** Fractional Belief Propagation $BP(\vec{c})$

1: initialize($m_{\alpha i}, Q_i \propto \prod_{N_i} m_{\alpha i}$)
2: **repeat**
3:    **for all** $\alpha$ **do**
4:       update $Q_\alpha$ according to (15).
5:       update $m_{\alpha i}, i \in N_\alpha$ according to (17) using the new $Q_\alpha$ and the old $Q_i$.
6:       update $Q_i, i \in N_\alpha$ by marginalization of $Q_\alpha$.
7:    **end for**
8: **until** convergence criterion is met (or maximum number of iterations is exceeded)
9: **return** $\{Q_\alpha, Q_i\}$ (or failure)

As a theoretical footnote we mention a different (generally more greedy) $\vec{c}$-algorithm, which has the same fixed points as $BP(\vec{c})$. This algorithm is similar to Algorithm 1, except that (1) the update of $Q_\alpha$ (in line 4) is to be taken with $c_\alpha = 1$, as in in standard belief propagation and (2) the update of the marginals $Q_i$ (in line 6) is to be performed by minimizing the divergence $D_{1/c_\alpha}(Q_\alpha, \prod_{i \in N_\alpha} Q_i)$ where

$$D_\epsilon(P, Q) = \frac{1}{\epsilon(1-\epsilon)}\left(1 - \sum_x P(x)^\epsilon Q(x)^{1-\epsilon}\right) , \qquad (18)$$

with the limiting cases

$$D_1(P, Q) = \sum_x P(x) \log \frac{P(x)}{Q(x)} \quad \text{and} \quad D_0(P, Q) = \sum_x Q(x) \log \frac{Q(x)}{P(x)} , \qquad (19)$$

rather than by marginalization (which corresponds to minimizing $D_1$, which is the equal to the usual $KL$ divergence). The $D_\epsilon$'s are known as the $\alpha$-divergences [9] where $\epsilon = \frac{1}{2}(\alpha+1)$ and $-1 \leq \alpha \leq 1$. The minimization of the $Q_i$'s using $D_0$ leads to the well known mean field equations.

# 6   Tuning $\vec{c}$ using linear response theory

Now the question is, how do we set the parameters $\vec{c}$? The idea is as follows, if we could have access to the true marginals $P_\alpha(x_\alpha) = P(x_\alpha)$, we could optimize $\vec{c}$ by minimizing, for example,

$$Cost(\vec{c}) = \sum_\alpha KL(P_\alpha, Q_\alpha^{\vec{c}}) \equiv \sum_\alpha \sum_{x_\alpha} P_\alpha(x_\alpha) \log \frac{P_\alpha(x_\alpha)}{Q_\alpha^{\vec{c}}(x_\alpha)} . \qquad (20)$$

in which we labeled $Q$ by $\vec{c}$ to emphasize its dependency on the scale parameters. Unfortunately, we do not have access to the true pair marginals, but if we would have estimates $\hat{P}_\alpha^{\vec{c}}$ that improve upon $Q_\alpha^{\vec{c}}$, we can compute new parameters $\vec{c}'$ such that $Q_\alpha^{\vec{c}'}$ is closer to $\hat{P}_\alpha^{\vec{c}}$. However, with the new parameters the estimates $\hat{P}_\alpha^{c'}$ will be changed as well, and this procedure should be iterated.

In this paper, we use the linear response theory [10] to improve upon $Q_\alpha^{\vec{c}}$. For simplicity, we restrict ourselves to Boltzmann machines with binary units. Applying linear response theory to $BP(\vec{c})$ in Boltzmann machines yields the following linear response estimates for the pair marginals,

$$Q_{ij}^{\vec{c}-\mathrm{LR}}(x_{ij}) = Q_i^{\vec{c}}(x_i)Q_j^{\vec{c}}(x_j) + \frac{x_j}{2}\frac{\partial Q_i^{\vec{c}}(x_i)}{\partial \theta_j} . \qquad (21)$$

**Algorithm 2** Tuning $\vec{c}$ by linear response

---

1: initialize($t = 1, \vec{c}_t = 1$)
2: **repeat**
3:     set step-size $\eta_t$
4:     compute the linear response estimates $Q_{ij}^{\vec{c}_t - \mathrm{LR}}$ as in (21)
5:     compute $\vec{c}_{t+1}$ as in (22).
6:     set $t = t + 1$
7: **until** convergence criterion is met
8: **return** $\left\{ Q_{ij}^{\vec{c}_t}, Q_i^{\vec{c}_t} \right\}$

---

In [10], it is argued that if $Q^{\vec{c}}(x_{ij})$ is correct up to $\mathcal{O}(\epsilon)$, the error in the linear response estimate is $\mathcal{O}(\epsilon^2)$. Linear response theory has been applied previously to improve upon pair marginals (or correlations) in the naive mean field approximation [11] and in loopy belief propagation [12].

To iteratively compute new scaling parameters from the linear response corrections we use a gradient descent like algorithm

$$\vec{c}_{t+1} = \vec{c}_t - \eta_t \nabla_{\vec{c}} \sum_{(ij)} KL(Q_{ij}^{\vec{c}_t - \mathrm{LR}}, Q_{ij}^{\vec{c}}) \tag{22}$$

with a time dependent step-size parameter $\eta_t$.

By iteratively computing the linear response marginals, and adapting the scale parameters in the gradient descent direction, we can optimize $\vec{c}$, see Algorithm 2. Each linear response estimate can be computed numerically by applying $BP(\vec{c})$ to a Boltzmann machine with parameters $(w, \theta)$ and $(w, \theta + \Delta\theta_j)$. Partial derivatives with respect to $c_{ij}$, required for the gradient in (22), can be computed numerically by rerunning fractional belief propagation with parameters $\vec{c} + \Delta c_{ij}$. In this procedure the computation cost to update $\vec{c}$ requires $\mathcal{O}(N) + \mathcal{O}(E)$ times the cost of $BP(\vec{c})$, where $N$ is the number of nodes and $E$ is the number of edges.

## 7 Numerical results

We applied the method to a Boltzmann machine in which the nodes are connected according to a $3 \times 3$ square grid with periodic boundary conditions. The weights in the model were drawn from the binary distribution $w_{ij} \in \{-0.5, 0.5\}$ with equal probability. Thresholds were drawn according to $\theta_i \sim \mathcal{N}(0, 0.1)$ We generated 20 networks, and compared results of standard loopy belief propagation to results obtained by fractional belief propagation where the scale parameters were obtained by Algorithm 2.

In the experiment the step size was set to be $\eta_t = 1/\log(1 + t)$. The iterations were stopped if the maximum change in $1/c_{ij}$ was less than $10^{-4}$, or if the number of iterations exceeded $t = 100$. Throughout the procedure, fractional belief propagations were ran with convergence criterion of maximal difference of $10^{-8}$ between messages in successive iterations (one iteration is one cycle over all weights). In our experiment, all (fractional) belief propagation runs converged. The number of updates of $\vec{c}$ ranged between 20 and 80. After optimization we found (inverse) scale parameters ranging from $1/c_{ij} \approx -0.5$ to $1/c_{ij} \approx 2$.

Results are plotted in figure 1. In the left panel, it can be seen that the procedure can lead to significant improvements. In these experiments, the solutions obtained by optimized $BP(\vec{c})$ are consistently 10 to 100 times better in averaged $KL$, than the ones obtained by

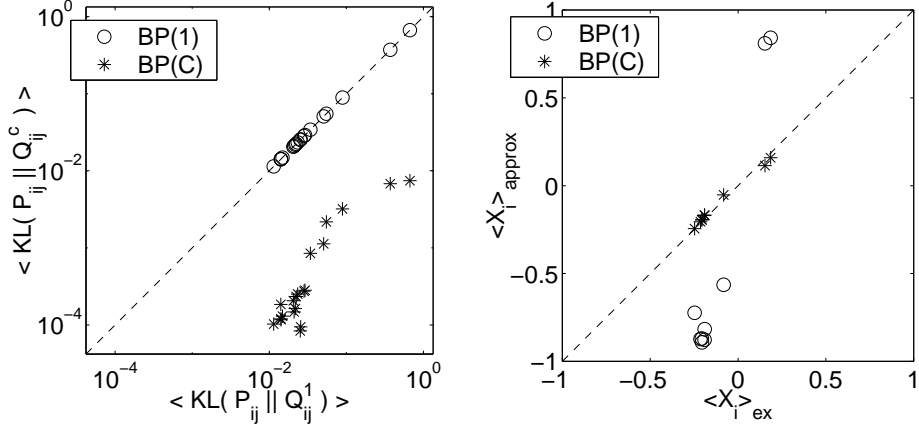

Figure 1: Left: Scatter plots of averaged $KL$ between exact and approximated pair marginals obtained by the optimized fractional belief propagation ($BP(\vec{c})$) versus the ones obtained by standard belief propagation ($BP(1)$). Each point in the plot is the result of one instantiation of the network. Right: approximated single-node means for $BP(1)$ and optimized $BP(\vec{c})$ against the exact single node means. This plot is for the network where $BP(1)$ had the worst performance (i.e. corresponding to the point in the left panel with highest $< KL(P_{ij}, Q_{ij}^1) >$).

standard $BP(1)$. The averaged $KL$ is defined as

$$\langle KL(P_{ij}, Q_{ij}) \rangle = \frac{1}{E} \sum_{(ij)} KL(P_{ij}, Q_{ij}) . \tag{23}$$

In the right panel, approximations of single-node means are plotted for the case where $BP(1)$ had the worst performance. Here we see that procedure can lead to quite precise estimates of the means, even if the quality of solutions by obtained $BP(1)$ is very poor. Here, it should be noticed that the linear response correction does not alter the estimated means [12]. In other words, the improvement in quality of the means is a result of optimized $\vec{c}$, and not of the linear response correction.

## 8 Conclusions

In this paper, we introduced fractional belief propagation as a family of approximating inference methods that generalize upon loopy belief propagation without resorting to larger clusters. The approximations are parameterized by scale parameters $c_\alpha$, which are motivated to better model the effective interactions due to the effect of loops in the graph. The approximations are formulated in terms of approximating free energies. This family of approximating free energies includes as special cases the Bethe free energy, the mean field free energy, and also the free energy approximation that provides an upper bound on the log partition function, developed in [7].

In order to apply fractional belief propagation, the scale parameters have to be tuned. In this paper, we demonstrated in toy problems for Boltzmann machines that it is possible to tune the scale parameters using linear response theory. Results show that considerable improvements can be obtained, even if standard loopy belief propagation is of poor quality. In principle, the method is applicable to larger and more general graphical models. However,

how to make the tuning of scale parameters practically feasible in such models is still to be explored.

## Acknowledgements

We thank Bert Kappen for helpful comments and the Dutch Technology Foundation STW for support.

## Footnotes

[1] $BP(1)$, i.e. with all $c_\alpha = 1$, is equivalent to standard loopy belief propagation

## References

[1] J. Pearl. *Probabilistic Reasoning in Intelligent systems: Networks of Plausible Inference*. Morgan Kaufmann Publishers, Inc., 1988.

[2] J. Yedidia, W. Freeman, and Y. Weiss. Generalized belief propagation. In *NIPS 13*.

[3] A. Yuille. CCCP algorithms to minimize the Bethe and Kikuchi free energies: Convergent alternatives to belief propagation. *Neural Computation*, July 2002.

[4] T. Minka. *A family of algorithms for approximate Bayesian inference*. PhD thesis, MIT Media Lab, 2001.

[5] S.L. Lauritzen and D.J. Spiegelhalter. Local computations with probabilties on graphical structures and their application to expert systems. *J. Royal Statistical society B*, 50:154–227, 1988.

[6] F. Kschischang, B. Frey, and H. Loeliger. Factor graphs and the sum-product algorithm. *IEEE Transactions on Information Theory*, 47(2):498–519, 2001.

[7] W. Wainwright, T. Jaakkola, and S. Willsky. A new class of upper bounds on the log partition function. In *UAI-2002*, pages 536–543.

[8] T. Heskes. Stable fixed points of loopy belief propagation are minima of the Bethe free energy. In *NIPS 15*.

[9] S. Amari, S. Ikeda, and H. Shimokawa. Information geometry of $\alpha$-projection in mean field approximation. In M. Opper and D. Saad, editors, *Advanced Mean Field Methods*, pages 241–258, Cambridge, MA, 2001. MIT press.

[10] G. Parisi. *Statistical Field Theory*. Addison-Wesley, Redwood City, CA, 1988.

[11] H.J. Kappen and F.B. Rodríguez. Efficient learning in Boltzmann Machines using linear response theory. *Neural Computation*, 10:1137–1156, 1998.

[12] M. Welling and Y.W. Teh. Propagation rules for linear response estimates of joint pairwise probabilities. 2002. Submitted.
